# Covariance Estimation for High Dimensional Data Vectors Using the Sparse Matrix Transform

**Guangzhi Cao**    **Charles A. Bouman**
School of Electrical and Computer Enigeering
Purdue University
West Lafayette, IN 47907
{gcao, bouman}@purdue.edu

## Abstract

Covariance estimation for high dimensional vectors is a classically difficult problem in statistical analysis and machine learning. In this paper, we propose a maximum likelihood (ML) approach to covariance estimation, which employs a novel sparsity constraint. More specifically, the covariance is constrained to have an eigen decomposition which can be represented as a sparse matrix transform (SMT). The SMT is formed by a product of pairwise coordinate rotations known as Givens rotations. Using this framework, the covariance can be efficiently estimated using greedy minimization of the log likelihood function, and the number of Givens rotations can be efficiently computed using a cross-validation procedure. The resulting estimator is positive definite and well-conditioned even when the sample size is limited. Experiments on standard hyperspectral data sets show that the SMT covariance estimate is consistently more accurate than both traditional shrinkage estimates and recently proposed graphical lasso estimates for a variety of different classes and sample sizes.

## 1   Introduction

Many problems in statistical pattern recognition and analysis require the classification and analysis of high dimensional data vectors. However, covariance estimation for high dimensional vectors is a classically difficult problem because the number of coefficients in the covariance grows as the dimension squared [1, 2]. This problem, sometimes referred to as the curse of dimensionality [3], presents a classic dilemma in statistical pattern analysis and machine learning.

In a typical application, one measures $n$ versions of a $p$ dimensional vector. If $n < p$, then the sample covariance matrix will be singular with $p - n$ eigenvalues equal to zero. Over the years, a variety of techniques have been proposed for computing a nonsingular estimate of the covariance. For example, regularized and shrinkage covariance estimators [4, 5, 6] are examples of such techniques.

In this paper, we propose a new approach to covariance estimation, which is based on constrained maximum likelihood (ML) estimation of the covariance [7]. In particular, the covariance is constrained to have an eigen decomposition which can be represented as a sparse matrix transform (SMT) [8, 9]. The SMT is formed by a product of pairwise coordinate rotations known as Givens rotations [10]. Using this framework, the covariance can be efficiently estimated using greedy minimization of the log likelihood function, and the number of Givens rotations can be efficiently computed using a cross-validation procedure. The estimator obtained using this method is always positive definite and well-conditioned even when the sample size is limited.

In order to validate our model, we perform experiments using a standard set of hyperspectral data [11], and we compare against both traditional shrinkage estimates and recently proposed graphical lasso estimates [12] for a variety of different classes and sample sizes. Our experiments show that,

for this example, the SMT covariance estimate is consistently more accurate. The SMT method also has a number of other advantages. It seems to be particularly good when estimating small eigenvalues and their associated eigenvectors. The cross-validation procedure used to estimate the SMT model order requires little additional computation, and the resulting eigen decomposition can be computed with very little computation (i.e. $\ll p^2$ operations).

## 2 Covariance estimation for high dimensional vectors

In the general case, we observe a set of $n$ vectors, $y_1, y_2, \cdots, y_n$, where each vector, $y_i$, is $p$ dimensional. Without loss of generality, we assume $y_i$ has zero mean. We can represent this data as the following $p \times n$ matrix

$$Y = [y_1, y_2, \cdots, y_n] \ . \tag{1}$$

If the vectors $y_i$ are identically distributed, then the sample covariance is given by

$$S = \frac{1}{n} Y Y^t \ , \tag{2}$$

and $S$ is an unbiased estimate of the true covariance matrix with $R = \mathrm{E}\left[y_i y_i^t\right] = \mathrm{E}[S]$.

While $S$ is an unbiased estimate of $R$ it is also singular when $n < p$. This is a serious deficiency since as the dimension $p$ grows, the number of vectors needed to estimate $R$ also grows. In practical applications, $n$ may be much smaller than $p$ which means that most of the eigenvalues of $R$ are erroneously estimated as zero.

A variety of methods have been proposed to regularize the estimate of $R$ so that it is not singular. Shrinkage estimators are a widely used class of estimators which regularize the covariance matrix by shrinking it toward some target structures [4, 5, 13]. Shrinkage estimators generally have the form $\hat{R} = \alpha D + (1 - \alpha)S$, where $D$ is some positive definite matrix. Some popular choices for $D$ are the identity matrix (or its scaled version) [5, 13] and the diagonal entries of $S$, i.e. $\mathrm{diag}(S)$ [5, 14]. In both cases, the shrinkage intensity $\alpha$ can be estimated using cross-validation or boot-strap methods.

Recently, a number of methods have been proposed for regularizing the estimate by making either the covariance or its inverse sparse [6, 12]. For example, the graphical lasso method enforces sparsity by imposing an $L_1$ norm constraint on the inverse covariance [12]. Banding or thresholding can also be used to obtain a sparse estimate of the covariance [15].

### 2.1 Maximum likelihood covariance estimation

Our approach will be to compute a constrained maximum likelihood (ML) estimate of the covariance $R$, under the modeling assumption that eigenvectors of $R$ may be represented as a sparse matrix transform (SMT) [8, 9]. To do this, we first decompose $R$ as

$$R = E \Lambda E^t \ , \tag{3}$$

where $E$ is the orthonormal matrix of eigenvectors and $\Lambda$ is the diagonal matrix of eigenvalues. Then we will estimate the covariance by maximizing the likelihood of the data $Y$ subject to the constraint that $E$ is an SMT. By varying the order, $K$, of the SMT, we may then reduce or increase the regularizing constraint on the covariance.

If we assume that the columns of $Y$ are independent and identically distributed Gaussian random vectors with mean zero and positive-definite covariance $R$, then the likelihood of $Y$ given $R$ is given by

$$p_R(Y) = \frac{1}{(2\pi)^{\frac{np}{2}}} \, |R|^{-\frac{n}{2}} \exp\left\{-\frac{1}{2}\mathrm{tr}\{Y^t R^{-1} Y\}\right\} \ . \tag{4}$$

The log-likelihood of $Y$ is then given by [7]

$$\log p_{(E,\Lambda)}(Y) = -\frac{n}{2}\mathrm{tr}\{\mathrm{diag}(E^t S E)\Lambda^{-1}\} - \frac{n}{2}\log|\Lambda| - \frac{np}{2}\log(2\pi) \ , \tag{5}$$

where $R = E \Lambda E^t$ is specified by the orthonormal eigenvector matrix $E$ and diagonal eigenvalue matrix $\Lambda$. Jointly maximizing the likelihood with respect to $E$ and $\Lambda$ then results in the ML estimates

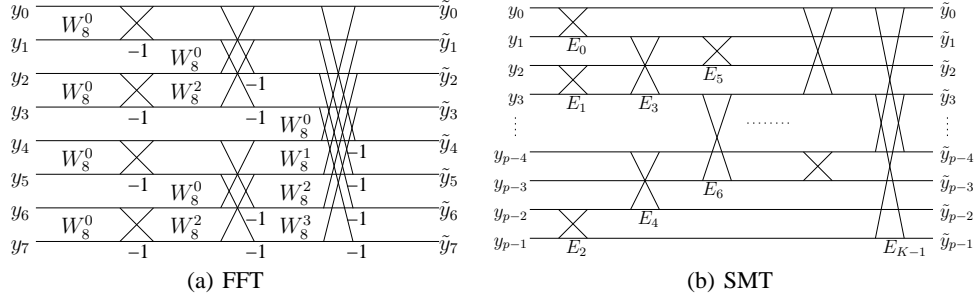

Figure 1: (a) 8-point FFT. (b) The SMT implementation of $\tilde{y} = Ey$. The SMT can be viewed as a generalization of FFT and orthonormal wavelet transforms.

of $E$ and $\Lambda$ given by [7]

$$\hat{E} = \arg\min_{E \in \Omega} \left\{ \left| \text{diag}(E^t S E) \right| \right\} \tag{6}$$

$$\hat{\Lambda} = \text{diag}(\hat{E}^t S \hat{E}) \,, \tag{7}$$

where $\Omega$ is the set of allowed orthonormal transforms. So we may compute the ML estimate by first solving the constrained optimization of (6), and then computing the eigenvalue estimates from (7).

## 2.2 ML estimation of eigenvectors using SMT model

The ML estimate of $E$ can be improved if the feasible set of eigenvector transforms, $\Omega$, can be constrained to a subset of all possible orthonormal transforms. By constraining $\Omega$, we effectively regularize the ML estimate by imposing a model. However, as with any model-based approach, the key is to select a feasible set, $\Omega$, which is as small as possible while still accurately modeling the behavior of the data.

Our approach is to select $\Omega$ to be the set of all orthonormal transforms that can be represented as an SMT of order $K$ [9]. More specifically, a matrix $E$ is an SMT of order $K$ if it can be written as a product of $K$ sparse orthornormal matrices, so that

$$E = \prod_{0}^{k=K-1} E_k = E_0 E_1 \cdots E_{K-1} \,, \tag{8}$$

where every sparse matrix, $E_k$, is a Givens rotation operating on a pair of coordinate indices $(i_k, j_k)$ [10]. Every Givens rotation $E_k$ is an orthonormal rotation in the plane of the two coordinates, $i_k$ and $j_k$, which has the form

$$E_k = I + \Theta(i_k, j_k, \theta_k) \,, \tag{9}$$

where $\Theta(i_k, j_k, \theta_k)$ is defined as

$$[\Theta]_{ij} = \begin{cases} \cos(\theta_k) - 1 & \text{if } i = j = i_k \text{ or } i = j = j_k \\ \sin(\theta_k) & \text{if } i = i_k \text{ and } j = j_k \\ -\sin(\theta_k) & \text{if } i = j_k \text{ and } j = i_k \\ 0 & \text{otherwise} \end{cases} \,. \tag{10}$$

Figure 1(b) shows the flow diagram for the application of an SMT to a data vector $y$. Notice that each 2D rotation, $E_k$, plays a role analogous to a "butterfly" used in a traditional fast Fourier transform (FFT) [16] in Fig. 1(a). However, unlike an FFT, the organization of the butterflies in an SMT is unstructured, and each butterfly can have an arbitrary rotation angle $\theta_k$. This more general structure allows an SMT to implement a larger set of orthonormal transformations. In fact, the SMT can be used to represent any orthonormal wavelet transform because, using the theory of paraunitary wavelets, orthonormal wavelets can be represented as a product of Givens rotations and delays [17]. More generally, when $K = \binom{p}{2}$, the SMT can be used to exactly represent any $p \times p$ orthonormal transformation [7].

Using the SMT model constraint, the ML estimate of $E$ is given by

$$\hat{E} = \arg \min_{E = \prod_0^{k=K-1} E_k} \left| \text{diag}(E^t S E) \right| . \tag{11}$$

Unfortunately, evaluating the constrained ML estimate of (11) requires the solution of an optimization problem with a nonconvex constraint. So evaluation of the globally optimal solutions is difficult. Therefore, our approach will be to use greedy minimization to compute a locally optimal solution to (11). The greedy minimization approach works by selecting each new butterfly $E_k$ to minimize the cost, while fixing the previous butterflies, $E_l$ for $l < k$.

This greedy optimization algorithm can be implemented with the following simple recursive procedure. We start by setting $S_0 = S$ to be the sample covariance, and initialize $k = 0$. Then we apply the following two steps for $k = 0$ to $K - 1$.

$$E_k^* = \arg \min_{E_k} \left| \text{diag}\left( E_k^t S_k E_k \right) \right| \tag{12}$$

$$S_{k+1} = E_k^{*t} S_k E_k^* . \tag{13}$$

The resulting values of $E_k^*$ are the butterflies of the SMT.

The problem remains of how to compute the solution to (12). In fact, this can be done quite easily by first determining the two coordinates, $i_k$ and $j_k$, that are most correlated,

$$(i_k, j_k) \leftarrow \arg \min_{(i,j)} \left( 1 - \frac{[S_k]_{ij}^2}{[S_k]_{ii}[S_k]_{jj}} \right) . \tag{14}$$

It can be shown that this coordinate pair, $(i_k, j_k)$, can most reduce the cost in (12) among all possible coordinate pairs [7]. Once $i_k$ and $j_k$ are determined, we apply the Givens rotation $E_k^*$ to minimize the cost in (12), which is given by

$$E_k^* = I + \Theta(i_k, j_k, \theta_k) , \tag{15}$$

where

$$\theta_k = \frac{1}{2} \text{atan}(-2[S_k]_{i_k j_k}, [S_k]_{i_k i_k} - [S_k]_{j_k j_k}) . \tag{16}$$

By iterating the (12) and (13) $K$ times, we obtain the constrained ML estimate of $E$ given by

$$\hat{E} = \prod_0^{k=K-1} E_k^* . \tag{17}$$

The model order, $K$, can be determined by a simple cross-validation procedure. For example, we can partition the data into three subsets, and $K$ is chosen to maximize the average likelihood of the left-out subsets given the estimated covariance using the other two subsets. Once $K$ is determined, the proposed covariance estimator is re-computed using all the data and the estimated model order.

The SMT covariance estimator obtained as above has some interesting properties. First, it is positive definite even for the limited sample size $n < p$. Also, it is permutation invariant, that is, the covariance estimator does not depend on the ordering of the data. Finally, the eigen decomposition $E^t y$ can be computed very efficiently by applying the $K$ sparse rotations in sequence.

## 2.3 SMT Shrinkage Estimator

In some cases, the accuracy of the SMT estimator can be improved by shrinking it towards the sample covariance. Let $\hat{R}_{SMT}$ represent the SMT covariance estimator. Then the SMT shrinkage estimate (SMT-S) can be obtained as

$$\hat{R}_{SMT-S} = \alpha \hat{R}_{SMT} + (1-\alpha)S , \tag{18}$$

where the parameter $\alpha$ can be computed using cross validation. Notice that

$$p_{\hat{R}_{SMT-S}}(Y) = p_{\hat{E}\hat{R}_{SMT-S}\hat{E}^t}(\hat{E}Y) = p_{\alpha\hat{\Lambda}+(1-\alpha)\hat{E}S\hat{E}^t}(\hat{E}Y) . \tag{19}$$

So cross validation can be efficiently implemented as in [5].

# 3 Experimental results

The effectiveness of the SMT covariance estimation depends on how well the SMT model can capture the behavior of real data vectors. Therefore in this section, we compare the performance of the SMT covariance estimator to commonly used shrinkage and graphical lasso estimators. We do this comparison using hyperspectral remotely sensed data as our high dimensional data vectors.

The hyperspectral data we use is available with the recently published book [11]. Figure 2(a) shows a simulated color IR view of an airborne hyperspectral data flightline over the Washington DC Mall. The sensor system measured the pixel response in 191 effective bands in the 0.4 to 2.4 $\mu$m region of the visible and infrared spectrum. The data set contains 1208 scan lines with 307 pixels in each scan line. The image was made using bands 60, 27 and 17 for the red, green and blue colors, respectively. The data set also provides ground truth pixels for five classes designated as grass, water, roof, street, and tree. In Fig. 2(a), the ground-truth pixels of the grass class are outlined with a white rectangle. Figure 2(b) shows the spectrum of the grass pixels, and Fig. 2(c) shows multivariate Gaussian vectors that were generated using the measured sample covariance for the grass class.

For each class, we computed the "true" covariance by using all the ground truth pixels to calculate the sample covariance. The covariance is computed by first subtracting the sample mean vector for each class, and then computing the sample covariance for the zero mean vectors. The number of pixels for the ground-truth classes of grass, water, roof, street, and tree are 1928, 1224, 3579, 416, and 388, respectively. In each case, the number of ground truth pixels was much larger than 191, so the true covariance matrices are nonsingular, and accurately represent the covariance of the hyperspectral data for that class.

## 3.1 Review of alternative estimators

A popular choice of the shrinkage target is the diagonal of $S$ [5, 14]. In this case, the shrinkage estimator is given by

$$\hat{R} = \alpha \text{diag}\,(S) + (1 - \alpha)\,S\,. \tag{20}$$

We use an efficient algorithm implementation of the leave-one-out likelihood (LOOL) cross-validation method to choose $\alpha$ as suggested in [5].

An alternative estimator is the graphic lasso (glasso) estimate recently proposed in [12] which is an $L_1$ regularized maximum likelihood estimate, such that

$$\hat{R} = \arg \max_{R \in \Psi} \left\{ \log(Y \mid R) - \rho \parallel R^{-1} \parallel_1 \right\}\,, \tag{21}$$

where $\Psi$ denotes the set of $p \times p$ positive definite matrices and $\rho$ the regularization parameter. We used the R code for glasso that is publically available online. We found cross-validation estimation of $\rho$ to be difficult, so in each case we manually selected the value of $\rho$ to minimize the Kullback-Leibler distance to the known covariance.

## 3.2 Gaussian case

First, we compare how different estimators perform when the data vectors are samples from an ideal multivariate Gaussian distribution. To do this, we first generated zero mean multivariate vectors with the true covariance for each of the five classes. Next we estimated the covariance using the four methods, the shrinkage estimator, glasso, SMT and SMT shrinkage estimation. In order to determine the effect of sample size, we also performed each experiment for a sample size of $n = 80$, 40, and 20, respectively. Every experiment was repeated 10 times.

In order to get an aggregate accessment of the effectiveness of SMT covariance estimation, we compared the estimated covariance for each method to the true covariance using the Kullback-Leibler (KL) distance [7]. The KL distance is a measure of the error between the estimated and true distribution. Figure 3(a)(b) and (c) show plots of the KL distances as a function of sample size for the four estimators. The error bars indicate the standard deviation of the KL distance due to random variation in the sample statistics. Notice that the SMT shrinkage (SMT-S) estimator is consistently the best of the four.

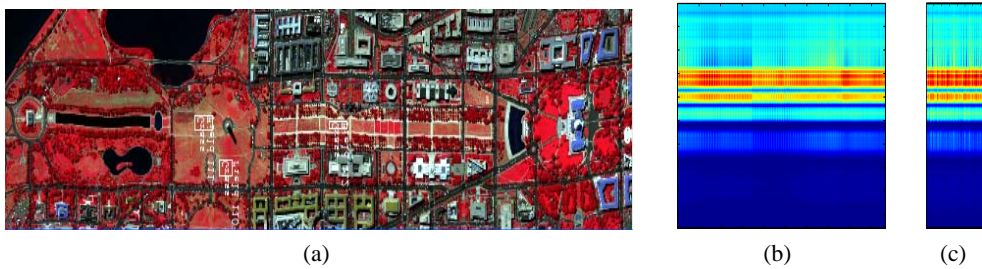

(a)  (b)  (c)

Figure 2: (a) Simulated color IR view of an airborne hyperspectral data over the Washington DC Mall [11]. (b) Ground-truth pixel spectrum of grass that are outlined with the white rectangles in (a). (c) Synthesized data spectrum using the Gaussian distribution.

Figure 4(a) shows the estimated eigenvalues for the grass class with $n = 80$. Notice that the eigenvalues of the SMT and SMT-S estimators are much closer to the true values than the shrinkage and glasso methods. Notice that the SMT estimators generate good estimates especially for the small eigenvalues.

Table 1 compares the computational complexity, CPU time and model order for the four estimators. The CPU time and model order were measured for the Guassian case of the grass class with $n = 80$. Notice that even with the cross validation, the SMT and SMT-S estimators are much faster than glasso. This is because the SMT transform is a sparse operator. In this case, the SMT uses an average of $K = 495$ rotations, which is equal to $K/p = 495/191 = 2.59$ rotations (or equivalently multiplies) per spectral sample.

### 3.3 Non-Gaussian case

In practice, the sample vectors may not be from an ideal multivariate Gaussian distribution. In order to see the effect of the non-Gaussian statistics on the accuracy of the covariance estimate, we performed a set of experiments which used random samples from the ground truth pixels as input. Since these samples are from the actual measured data, their distribution is not precisely Gaussian. Using these samples, we computed the covariance estimates for the five classes using the four different methods with sample sizes of $n = 80, 40$, and $20$.

Plots of the KL distances for the non-Gaussian grass case[1] are shown in Fig. 3(d)(e) and (f); and Figure 4(b) shows the estimated eigenvalues for grass with $n = 80$. Note that the results are similar to those found for the ideal Guassian case.

## 4  Conclusion

We have proposed a novel method for covariance estimation of high dimensional data. The new method is based on constrained maximum likelihood (ML) estimation in which the eigenvector transformation is constrained to be the composition of $K$ Givens rotations. This model seems to capture the essential behavior of the data with a relatively small number of parameters. The constraint set is a $K$ dimensional manifold in the space of orthonormal transforms, but since it is not a linear space, the resulting ML estimation optimization problem does not yield a closed form global optimum. However, we show that a recursive local optimization procedure is simple, intuitive, and yields good results.

We also demonstrate that the proposed SMT covariance estimation methods substantially reduce the error in the covariance estimate as compared to current state-of-the-art estimates for a standard hyperspectral data set. The MATLAB code for SMT covariance estimation is available at: https://engineering.purdue.edu/~bouman/publications/pub_smt.html.

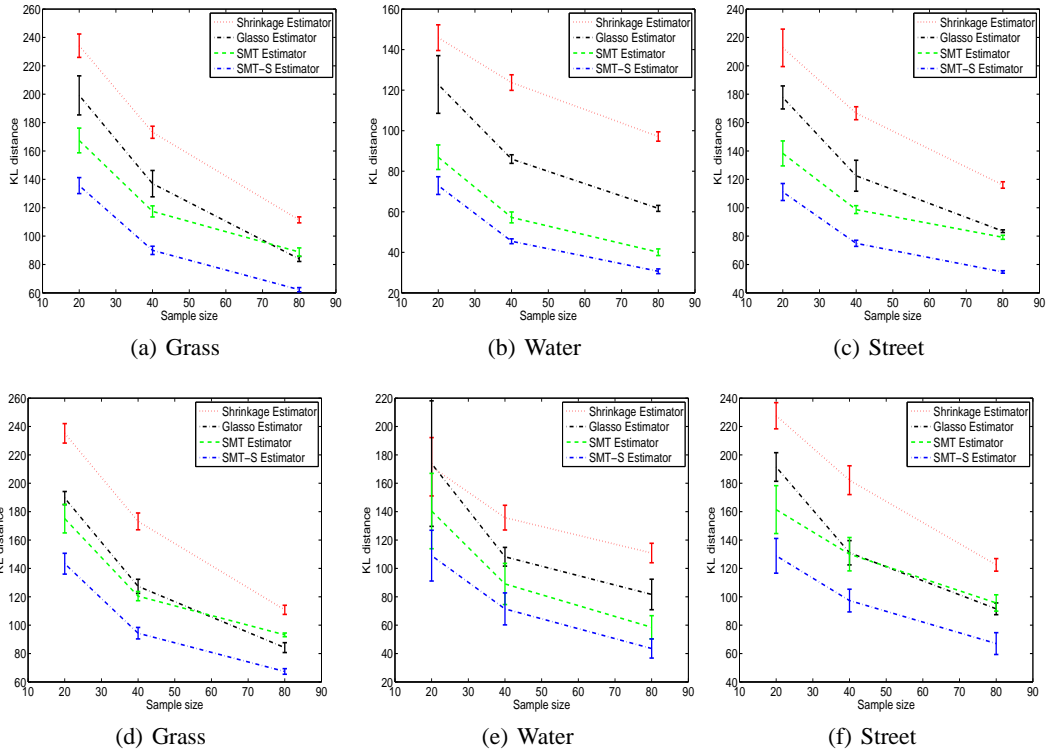

Figure 3: Kullback-Leibler distance from true distribution versus sample size for various classes: (a) (b) (c) Gaussian case (d) (e) (f) non-Gaussian case.

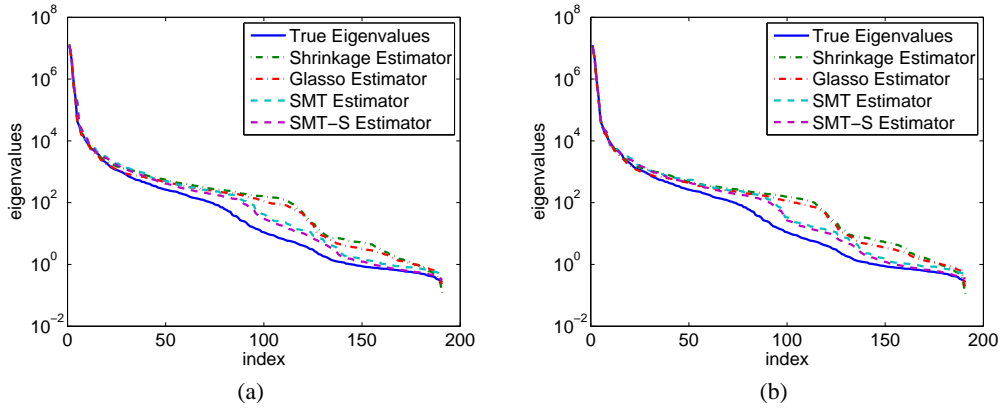

Figure 4: The distribution of estimated eigenvalues for the grass class with $n = 80$: (a) Gaussian case (b) Non-Gaussian case.

| | Complexity (without cross-validation) | CPU time (seconds) | Model order |
|---|---|---|---|
| Shrinkage Est. | $p$ | 8.6 (with cross-validation) | 1 |
| glasso | $p^3 I$ | 422.6 (without cross-validation) | 4939 |
| SMT | $p^2 + Kp$ | 6.5 (with cross-validation) | 495 |
| SMT-S | $p^2 + Kp$ | 7.2 (with cross-validation) | 496 |

Table 1: Comparison of computational complexity, CPU time and model order for various covariance estimators. The complexity is without cross validation and does not include the computation of the sample covariance (order of $np^2$). The CPU time and model order were measured for the Guassian case of the grass class with $n = 80$. $I$ is the number of cycles used in glasso.

**Acknowledgments**

This work was supported by the National Science Foundation under Contract CCR-0431024. We would also like to thank James Theiler (J.T.) and Mark Bell for their insightful comments and suggestions.

## Footnotes

[1] In fact, these are the KL distances between the estimated covariance and the sample covariance computed from the full set of training data, under the assumption of a multivariate Gaussian distribution.

# References

[1] C. Stein, B. Efron, and C. Morris, "Improving the usual estimator of a normal covariance matrix," Dept. of Statistics, Stanford University, Report 37, 1972.

[2] K. Fukunaga, *Introduction to Statistical Pattern Recognition*.   Boston, MA: Academic Press, 1990, 2nd Ed.

[3] A. K. Jain, R. P. Duin, and J. Mao, "Statistical pattern recognition: A review," *IEEE Transactions on Pattern Analysis and Machine Intelligence*, vol. 22, no. 1, pp. 4–37, 2000.

[4] J. H. Friedman, "Regularized discriminant analysis," *Journal of the American Statistical Association*, vol. 84, no. 405, pp. 165–175, 1989.

[5] J. P. Hoffbeck and D. A. Landgrebe, "Covariance matrix estimation and classification with limited training data," *IEEE Transactions on Pattern Analysis and Machine Intelligence*, vol. 18, no. 7, pp. 763–767, 1996.

[6] P. J. Bickel and E. Levina, "Regularized estimation of large covariance matrices," *Annals of Statistics*, vol. 36, no. 1, pp. 199–227, 2008.

[7] G. Cao and C. A. Bouman, "Covariance estimation for high dimensional data vectors using the sparse matrix transform," Purdue University, Technical Report ECE 08-05, 2008.

[8] G. Cao, C. A. Bouman, and K. J. Webb, "Fast reconstruction algorithms for optical tomography using sparse matrix representations," in *Proceedings of 2007 IEEE International Symposium on Biomedical Imaging*, April 2007.

[9] ——, "Non-iterative MAP reconstruction using sparse matrix representations," *(submitted to) IEEE Trans. on Image Processing*.

[10] W. Givens, "Computation of plane unitary rotations transforming a general matrix to triangular form," *Journal of the Society for Industrial and Applied Mathematics*, vol. 6, no. 1, pp. 26–50, March 1958.

[11] D. A. Landgrebe, *Signal Theory Methods in Multispectral Remote Sensing*.  New York: Wiley-Interscience, 2005.

[12] J. Friedman, T. Hastie, and R. Tibshirani, "Sparse inverse covariance estimation with the graphical lasso," *Biostatistics*, vol. 9, no. 3, pp. 432–441, Jul. 2008.

[13] M. J. Daniels and R. E. Kass, "Shrinkage estimators for covariance matrices," *Biometrics*, vol. 57, no. 4, pp. 1173–1184, 2001.

[14] J. Schafer and K. Strimmer, "A shrinkage approach to large-scale covariance matrix estimation and implications for functional genomics," *Statistical Applications in Genetics and Molecular Biology*, vol. 4, no. 1, 2005.

[15] P. J. Bickel and E. Levina, "Covariance regularization by thresholding," Department of Statistics, UC Berkeley, Technical Report 744, 2007.

[16] J. W. Cooley and J. W. Tukey, "An algorithm for the machine calculation of complex Fourier series," *Mathematics of Computation*, vol. 19, no. 90, pp. 297–301, April 1965.

[17] A. Soman and P. Vaidyanathan, "Paraunitary filter banks and wavelet packets," *Acoustics, Speech, and Signal Processing, 1992. ICASSP-92., 1992 IEEE International Conference on*, vol. 4, pp. 397–400 vol.4, Mar 1992.

